# A neural model of visual contour integration

Zhaoping Li
Computer Science, Hong Kong University of Science and Technology
Clear Water Bay, Hong Kong
zhaoping@uxmail.ust.hk[1]

## Abstract

We introduce a neurobiologically plausible model of contour integration from visual inputs of individual oriented edges. The model is composed of interacting excitatory neurons and inhibitory interneurons, receives visual inputs via oriented receptive fields (RFs) like those in V1. The RF centers are distributed in space. At each location, a finite number of cells tuned to orientations spanning $180°$ compose a model hypercolumn. Cortical interactions modify neural activities produced by visual inputs, selectively amplifying activities for edge elements belonging to smooth input contours. Elements within one contour produce synchronized neural activities. We show analytically and empirically that contour enhancement and neural synchrony increase with contour length, smoothness and closure, as observed experimentally. This model gives testable predictions, and in addition, introduces a feedback mechanism allowing higher visual centers to enhance, suppress, and segment contours.

## 1. Introduction

The visual system must group local elements in its input into meaningful global features to infer the visual objects in the scene. Sometimes local features group into regions, as in texture segmentation; at other times they group into contours which may represent object boundaries. Although much is known about the processing steps that extract local features such as oriented input edges, it is still unclear how local features are grouped into global ones more meaningful for objects. In this

study, we model the neural mechanisms underlying the grouping of edge elements into contours — contour integration.

Recent psychophysical and physiological observations[14, 8] demonstrate a decrease in detection threshold of an edge element, by human observers or a primary cortical cell, if there are aligned neighboring edge elements. Changes in neural responses by visual stimuli presented outside their RFs have been observed physiologically[9, 8]. Human observers easily identify a smooth curve composed of individual, even disconnected, Gabor "edge" elements distributed among many similar elements scattered in the background[4]. Horizontal neural connections observed in the primary visual cortex[5], and the finding that these connections preferably link cells tuned to similar orienations[5], provide a likely neural basis underlying the primitive visual grouping phenomena such as contour integration. These findings suggest that simple and local neural interactions even in V1 could contribute to grouping.

However, it has been difficult to model contour integration using only V1 elements and operation. Most existing models[15, 18] of contour integration lack well-founded biological bases. More neurally based models, e.g., the one by Grossberg and Mingolla[7], require operations beyond V1 or biologically questionable. It is thus desirable to find out whether contour enhancement can indeed occur within V1 or has to be attributed to top-down feedback. We introduce a V1 model of contour integration, using orientation selective cells, local cortical circuits, and horizontal connections. This model captures the essentials of the contour integration behavior. More details of the model can be found in a longer paper[12].

## 2. The Model

### 2.1 Model outline

$K$ neuron pairs at each spatial location $i$ model a hypercolumn in V1 (figure 1). Each neuron has a receptive field center $i$ and an optimal orientation $\theta = k\pi/K$ for $k = 1, 2, ...K$. A neuron pair consist of a connected excitatory neuron and inhibitory neuron which are denoted by indice $(i\theta)$ for their receptive field center and preferred orientation, and are referred to as an edge segment. An edge segment receives the visual input via the excitatory cell, whose output quantifies the saliency of the edge segment and projects to higher visual centers. The inhibitory cells are treated as interneurons. When an input image contains an edge at $i$ oriented at $\theta_o$, the edge segment $i\theta$ receives input $I_{i\theta} \propto \phi(\theta - \theta_o)$, where $\phi(\theta) = e^{-|\theta|/(\pi/8)}$ is a cell's orientation tuning curve.

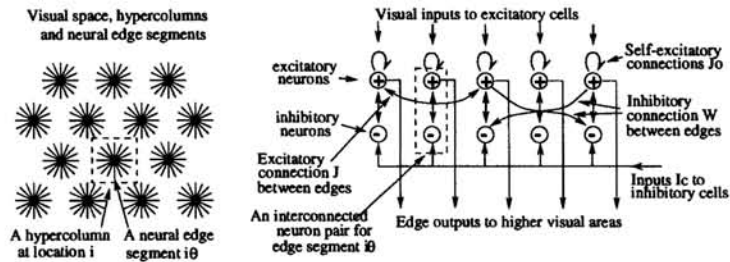

Figure 1: Model visual space, hypercolumn, edge segments, neural elements, visual inputs, and neural connections. The input space is a discrete hexagonal or Manhatten grid.

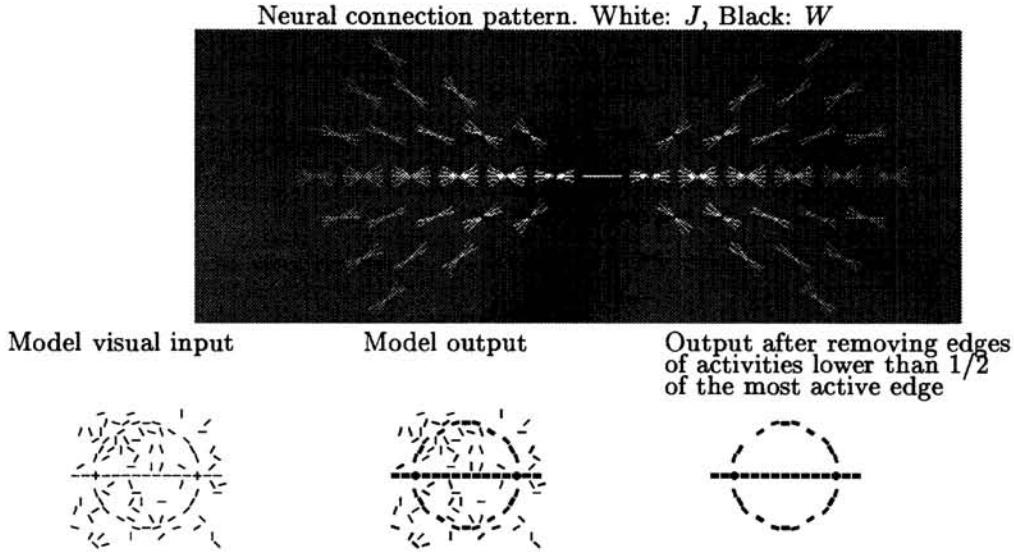

Figure 2: Model neural connections and performance for contour enhancement and noise reduction. The top graph depicts connections $J_{i\theta,j\theta'}$ and $W_{i\theta,j\theta'}$ respectively between the center (white) horizontal edge and other edges. The whiteness and blackness of each edge is proportional to the connection sthength $J_{i\theta,j\theta'}$ and $W_{i\theta,j\theta'}$ respectively. The bottom row plots visual input and the maximum model outputs. The edge thickness in this row is proportional to the value of edge input or activity. The same format applies to the other figures in this paper.

Let $x_{i\theta}$ and $y_{i\theta}$ be the cell membrane potentials for the excitatory and inhibitory cells respectively in the edge segment, then

$$\dot{x}_{i\theta} = -\alpha_x x_{i\theta} - g_y(y_{i\theta}) + J_o g_x(x_{i\theta}) + \sum_{j\theta' \neq i\theta} J_{i\theta j\theta'} g_x(x_{j\theta'}) + I_{i\theta} + I_o \quad (1)$$

$$\dot{y}_{i\theta} = -\alpha_y y_{i\theta} + g_x(x_{i\theta}) + \sum_{j\theta' \neq i\theta} W_{i\theta j\theta'} g_x(x_{j\theta'}) + I_c \quad (2)$$

where $g_x(x_{i\theta})$ and $g_y(y_{i\theta})$ are the firing rates from the excitatory and inhibitory cells respectively, $1/\alpha_x$ and $1/\alpha_y$ are the membrane constants, $J_o$ is the self-excitatory connection weight, $J_{i\theta,j\theta'}$ and $W_{i\theta,j\theta'}$ are synaptic weights between neurons, and $I_o$ and $I_c$ are the background inputs to the excitatory and inhibitory cells. Without loss of generality, we take $\alpha_x = \alpha_y = 1$, $g_x(x)$ as threshold linear with saturation and an unit gain $g'_x(x) = 1$ in the linear range.

The synaptic connections $J_{i\theta,j\theta'}$ and $W_{i\theta,j\theta'}$ are local and translation and rotation invariant (Figure (2)). $J_{i\theta,j\theta'}$ increases with the smoothness (small curvature) of the curve that best connects $(i\theta)$ and $(j\theta')$, and edge elements inhibit each other via $W_{i\theta,j\theta'}$ when they are alternative choices in a smooth curve route. Given an input pattern $I_{i\theta}$, the network approaches a dynamic state after several membrane time constants. As in Figure (2), the neurons with relatively higher final activities are those belonging to smooth curves in the input.

## 2.2 Model analysis

Ignoring neural connections between edge segments, the neuron in edge segment $i\theta$

has input sensitivity

$$\delta g_x(x_o)/\delta I_{i\theta} \quad = \quad \frac{g'_x(x_o)}{1 + g'_y(y_o)g'_x(x_o) - J_o g'_x(x_o)} \qquad (3)$$

$$\delta g_x(x_o)/\delta I_c \quad = \quad -\frac{g'_y(y_o)g'_x(x_o)}{1 + g'_y(y_o)g'_x(x_o) - J_o g'_x(x_o)} \qquad (4)$$

where $g_x(x_o)$ and $g_y(y_o)$ are roughly the average neural activities (omitting $i\theta$ for simplicity). Thus the edge activity increases with $I_{i\theta}$ and decreases with $I_c$ (in cases that interest us, $g'_y(y_o)g'_x(x_o) > J_o g'_x(x_o) - 1$). The resulting input-output function given $I_c$, $g_x(x_o)$ vs. $I_{i\theta}$, corresponds well with physiological data.

By effectively increasing $I_{i\theta}$ or $I_c$, the edge element $(j\theta')$ can excite or inhibit the element $(i\theta)$ with excitatory-to-excitatory input $J_{i\theta,j\theta'}g_x(x_{j\theta'})$ and excitatory-to-inhibitory input $W_{i\theta j\theta'}g_x(x_{j\theta'})$ respectively. Contour enhancement is so (Fig. 2) achieved. In the simplest example when the visual input has equally spaced equal strength edges from a line and all other edge segments are silent, we can treat the line system as one dimensional, omit $\theta$ and take $i$ as locations along the line. A lack of inhibition between line segments gives:

$$\dot{x}_i \quad = \quad -\alpha_x x_i - g_y(y_i) + J_o g_x(x_i) + \sum_{j\neq i} J_{ij}g_x(x_j) + I_o + I_{line-input} \qquad (5)$$

$$\dot{y}_i \quad = \quad -\alpha_y y_i + g_x(x_i) + I_c \qquad (6)$$

If line is infinite, by symmetry, each edge segment has the same average activity $g_x(x_i) \sim g_x(x_o)$ for all $i$. This system can then be seen[12] either as a giant edge with self-excitatory connection $(J_o + \sum_{j\neq i} J_{ij})$, or a single edge with extra external input $\Delta I = (\sum_{j\neq i} J_{ij})g_x(x_o)$. Either way, activities $g_x(x_o)$ are enhanced for each edge element in the line (figure 3 and 2).

This analysis is also applicable to constant curvature curves[12]. It can be shown that, in the linear range of $g_x()$, the response ratio between a curve segment and an isolated segment is $(g'_y(y_o) + 1 - J_o)/(g'_y(y_o) + 1 - J_o - \sum_{i\neq j} J_{ij})$. Since $\sum_{i\neq j} J_{ij}$ decreases with increasing curvature, so does the response enhancement. Translation invariance along the curve breaks down in a finite length curve near its two ends, where activity enhancement decays by a decreased excitation from fewer neighboring segments. This suggests that a closed or longer curve has higher saliency than an open or shorter one (figure (3)). This prediction is expected to hold also for curves of non-constant curvature, and should play a significant role in the psychophysical observation[10] showing a decreased detection threshold for closed curves from that of the open ones.

Further analysis[12] shows that the edge segments in a curve normally exhibit neural oscillations around their mean activity levels with near-zero phase delays from each other. The model predicts that, like the contour enhancement, the oscillation is stronger for longer, smoother, and closed curves than open and shorter ones, and tapers off near curve endings where oscillation synchrony also deteriorates (figure (3)).

## 2.3 Central feedback control for contour enhancement, supression, filling in, and segmentation

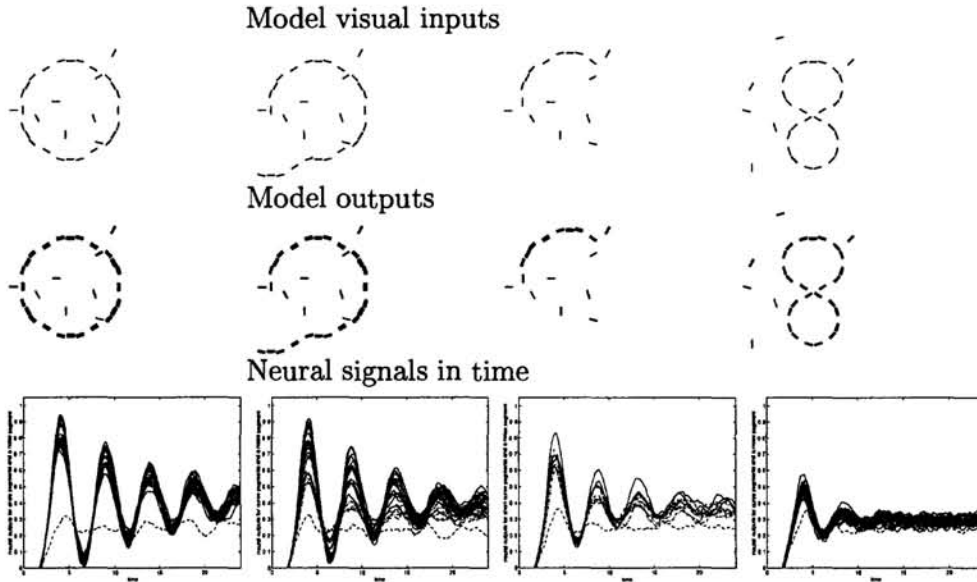

Figure 3: Model performance for input curves and noises. Each column is dedicated to one input condition. The top row is the visual input; the middle row is the maximum neural responses, and the bottom row the segment outputs as a function of time. The neural signals in the bottom row are shown superposed. The solid curves plot outputs for the segments away from the curve endings, the dash-dotted curves for segments near the curve endings, and the dashed curve in each plot, usually the lowest lying one, is an example from a noise segment. Note the decrease in neural signal synchrony between the curve segments, in the oscillation amplitudes, and average neural activities for the curve segments, as the curve becomes open, shorter, and more curled or when the segment is near the curve endings. Because the model employs discrete a grid input space, the figure '8' curve in the right column is almost not a smooth curve, hence the contour is only very weakly enhanced. The enhancement for this curve could be increased by a denser input grid or a multiscale input space.

The model assumes that higher visual centers send inputs $I_c$ to the inhibitory cells, to influence neural activities by equation (4). Take $I_c = I_{c,background} + I_{c,control} \geq 0$, where $I_{c,background}$ is the same for all edge segments and is useful for modulating the overall visual alertness level. By setting $I_{c,control}$ differently for different edge segments, higher centers can selectively suppress or enhance activities for some visual objects (contours) (compare figure 4D,G), and even effectively achieve contour segmentation (figure (4H)) by silencing segments from a given curve. A similar feedback control mechanism was used in an olfactory model for odor sensitivity control and odor segmentation[11]. Nevertheless, the feedback cannot completely substitute for the visual input $I_{i\theta}$. By equation (4), $I_c$ is effective only when $g_y'(y_o)g_x'(x_o) \neq 0$. With insufficient background input $I_o$, some visual input or excitation from other segments is needed to have $g_x'(x_o) > 0$, even when all the inhibition from $I_c$ is removed by central control. However, a segment with weak visual input or excitation from aligned neighboring segments can increase its activity or become active from subthreshold. Therefore, this model central feedback can enhance a weak contour or fill in an incomplete one (compare figure 4F,I), but can not enhance or "hallucinate" any contour not existing at least partially in the visual input $I$ (figure 4E).

## 3. Summary and Discussion

We have presented a model which accounts for phenomena of contour enhance-

**A: Input**          **D: Output for A**          **G: Output for A, line suppresion**
                      no central control           circle enhancement

**B: Input**          **E: Output for B, enhancement**   **H: Same as G, except with**
                      for circle and non-existing line    stronger line suppresion

**C: Input**          **F: Output for C**          **I: Output for C, enhancement**
                      no central control           for both line and circle

Figure 4: Central feedback control. The visual inputs are in **A, B** and **C**. The rest
are the model responses under different feedback conditions. **D:** model response for
input **A** without central control. **G:** model response for input **A** with line suppres-
sion by $I_{c,control} = 0.25 I_{c,background}$ on the line segments, and circle enhancement by
$I_{c,control} = -0.29 I_{c,background}$ on the circle elements. **H:** Same as **G** except the line sup-
pression signal $I_{c,control}$ is doubled. **E:** Model response to input **B** with enhancement for
the line and the circle by central feedback $I_{c,control} = -0.29 I_{c,background}$. **F:** Response to
input **C** with no central control. **I:** Response to input **C** with line and circle enhancement
by $I_{c,control} = -0.29 I_{c,background}$. Note how the input gaps are partiallly filled in **F** and
almost completely filled in **I**. Note that the apparent gaps in the circle are caused by the
underlying discrete grid in the model input space, and hence no gap actually exists, and
no filling in is needed, for this circle. Also, with the wrap around boundary condition, the
line is actually a closed or infinitely long line, and is thus naturally more salient in this
model without feedback.

ment using neurally plausible elements in V1. It is shown analytically and empiri-
cally that both the contour enhancement and the neural oscillation amplitudes are
stronger for longer, closed, and smaller curvature curves, agreeing with experimental
observations[8, 4, 10, 6, 2]. The model predicts that horizontal connections target
preferentially excitatory or inhibitory post-synaptic cells when the linked edges are
aligned or less aligned (Fig. 2). In addition, we introduce a possible feedback mech-
anism by which higher visual centers could selectively enhance or suppress contour
activities, and achieve contour segmentation. This feedback mechanism has the de-
sirable property that while the higher centers can enhance or complete an existing
weak and/or fragmented input contour, they cannot enhance a non-existant contour
in the input, thus preventing "hallucination". This property could be exploited by
higher visual centers for hypothesis testing and object reconstruction by cooperat-
ing with lower visual centers. Analogous computational mechanisms have been used
in an olfactory model to achieve odor segmentation and sensitivity modulation[11].
It will be interesting to explore the universality of such computational mechanisms
across sensory modalities.

The organization of the model is based on various experimental finding[17, 5, 1,
8, 14]: recurrent excitatory-inhibitory interactions; excitatory and inhibitory link-
ing of edge elements with similar orientation preferences; and neural connection

patterns. At the cost of analytical tractability without essential changes in model performance, one can relax the model's idealization of a 1:1 ratio in the excitatory and inhibitory cell numbers, the lack of connections between the inhibitory cells, and the excitatory cells as the exclusive recipients of visual input. While abundant feedback connections are observed from higher visual centers to the primary visual cortex, there is as yet no clear indication of cell types of their targets[3, 16]. It is desirable to find out whether the feedback is indeed directed to the inhibitory cells as predicted.

This model can be extended to stereo, temporal, and chromatic dimensions, by linking edge segments aligned in orientation, depth, motion direction and color. V1 cells have receptive field tuning in all these dimensions, and cortical connections are indeed observed to link cells of similar receptive field properties[5]. This model does not model many other apparently non-contour related visual phenomena such as receptive field adaptations[5]. It is also beyond this model to explain how the higher visual centers decide which segments belong to one contour in order to achieve feedback control, although it has been hypothesized that phase locked neural oscillations and neural correlations can play such a role[13].

## Footnotes

[1]I would very much like to thank Jochen Braun for introducing me to the topic, and Peter Dayan for many helpful conversations and comments on the drafts. This work was supported by the Hong Kong Research Grant Council.

# References

[1] Douglas R.J. and Martin K. A. "Neocortex" in *Synaptic Organization of the Brain* 3rd Edition, Ed. G. M. Shepherd, Oxford University Press 1990.

[2] Eckhorn R, Bauer, R. Jordan W. Brosch, M. Kruse W., Munk M. and Reitboeck, H. J. 1988. *Biol. Cybern.* 60:121-130.

[3] van Essen D. Peters and E G Jones, Plenum Press, New York, 1985. p. 259-329

[4] Field DJ; Hayes A; Hess RF 1993. *Vision Res.* 1993 Jan; 33(2): 173-93

[5] Gilbert CD *Neuron.* 1992 Jul; 9(1): 1-13

[6] Gray C.M. and Singer W. 1989 *Proc. Natl. Acad. Sci. USA* 86: 1698-1702.

[7] Grossberg S; Mingolla E *Percept Psychophys.* 1985 Aug; 38(2): 141-71

[8] Kapadia MK; Ito M; Gilbert CD; Westheimer G, *Neuron.* 1995 Oct; 15(4): 843-56

[9] Knierim J. J. and van Essen D. C. 1992, *J. Neurophysiol.* 67, 961-980.

[10] Kovacs I; Julesz B *Proc Natl Acad Sci USA.* 1993 Aug 15; 90(16): 7495-7

[11] Li Zhaoping, *Biological Cybernetics*, 62/4 (1990), P. 349-361

[12] Li Zhaoping, "A neural model contour integration in primary visual cortex" manuscript submitted for publication.

[13] von der Malsburg C. 1981 "The correlation theory of brain function." Internal report, Max-Planck-Institute for Biophysical Chemistry, Gottingen, West Germany.

[14] Polat U; Sagi D, 1994 *Vision Res.* 1994 Jan; 34(1): 73-8

[15] Shashua A. and Ullman S. 1988 *Proceedings of the International Conference on Computer Vision.* Tempa, Florida, 482-488.

[16] Valverde F. in *Cerebral Cortex* Eds. A Peters and E G Jones, Plenum Press, New York, 1985. p. 207-258.

[17] White E. L. *Cortical circuits* 46-82, Birkhauser, Boston, 1989

[18] Zucker S. W., David C., Dobbins A, and Iverson L. in *Second international conference on computer vision* pp. 568-577, IEEE computer society press, 1988.
